# Variational EM Algorithms for Non-Gaussian Latent Variable Models

**J. A. Palmer,  D. P. Wipf,  K. Kreutz-Delgado,  and B. D. Rao**
Department of Electrical and Computer Engineering
University of California San Diego, La Jolla, CA 92093
{japalmer,dwipf,kreutz,brao}@ece.ucsd.edu

## Abstract

We consider criteria for variational representations of non-Gaussian latent variables, and derive variational EM algorithms in general form. We establish a general equivalence among convex bounding methods, evidence based methods, and ensemble learning/Variational Bayes methods, which has previously been demonstrated only for particular cases.

## 1   Introduction

Probabilistic methods have become well-established in the analysis of learning algorithms over the past decade, drawing largely on classical Gaussian statistical theory [21, 2, 28]. More recently, variational Bayes and ensemble learning methods [22, 13] have been proposed. In addition to the evidence and VB methods, variational methods based on convex bounding have been proposed for dealing with non-gaussian latent variables [18, 14]. We concentrate here on the theory of the linear model, with direct application to ICA [14], factor analysis [2], mixture models [13], kernel regression [30, 11, 32], and linearization approaches to nonlinear models [15]. The methods can likely be applied in other contexts.

In Mackay's evidence framework, "hierarchical priors" are employed on the latent variables, using Gamma priors on the inverse variances, which has the effect of making the marginal distribution of the latent variable prior the non-Gaussian Student's $t$ [30]. Based on Mackay's framework, Tipping proposed the Relevance Vector Machine (RVM) [30] for estimation of sparse solutions in the kernel regression problem. A relationship between the evidence framework and ensemble/VB methods has been noted in [22, 6] for the particular case of the RVM with $t$ hyperprior. Figueiredo [11] proposed EM algorithms based on hyperprior representations of the Laplacian and Jeffrey's priors. In [14], Girolami employed the convex variational framework of [16] to derive a different type of variational EM algorithm using a convex variational representation of the Laplacian prior. Wipf et al. [32] demonstrated the equivalence between the variational approach of [16, 14] and the evidence based RVM for the case of $t$ priors, and thus via [6], the equivalence of the convex variational method and the ensemble/VB methods for the particular case of the $t$ prior.

In this paper we consider these methods from a unifying viewpoint, deriving algorithms in more general form and establishing a more general relationship among the methods than has previously been shown. In §2, we define the model and estimation problems we shall be concerned with, and in §3 we discuss criteria for variational representations. In §4 we consider the relationships among these methods.

## 2 The Bayesian linear model

Throughout we shall consider the following model,

$$\mathbf{y} = \mathbf{A}\mathbf{x} + \nu \,, \tag{1}$$

where $\mathbf{A} \in \mathbb{R}^{m \times n}$, $\mathbf{x} \sim p(\mathbf{x}) = \prod_i p(x_i)$, and $\nu \sim \mathcal{N}(\mathbf{0}, \mathbf{\Sigma}_\nu)$, with $\mathbf{x}$ and $\nu$ independent. The important thing to note for our purposes is that the $x_i$ are non-Gaussian.

We consider two types of variational representation of the non-Gaussian priors $p(x_i)$, which we shall call *convex type* and *integral type*. In the convex type of variational representation, the density is represented as a supremum over Gaussian functions of varying scale,

$$p(x) = \sup_{\xi > 0} \mathcal{N}(x; 0, \xi^{-1}) \, \varphi(\xi) \,. \tag{2}$$

The essential property of "concavity in $x^2$" leading to this representation was used in [29, 17, 16, 18, 6] to represent the Logistic link function. A convex type representation of the Laplace density was applied to learning overcomplete representations in [14].

In the integral type of representation, the density $p(x)$ is represented as an integral over the scale parameter of the density, with respect to some positive measure $\mu$,

$$p(x) = \int_0^\infty \mathcal{N}(x; 0, \xi^{-1}) \, d\mu(\xi) \,. \tag{3}$$

Such representations with a general kernel are referred to as scale mixtures [19]. Gaussian scale mixtures were discussed in the examples of Dempster, Laird, and Rubin's original EM paper [9], and treated more extensively in [10]. The integral representation has been used, sometimes implicitly, for kernel-based estimation [30, 11] and ICA [20]. The distinction between MAP estimation of components and estimation of hyperparameters has been discussed in [23] and [30] for the case of Gamma distributed inverse variance.

We shall be interested in variational EM algorithms for solving two basic problems, corresponding essentially to the two methods of handling hyperparameters discussed in [23]: the MAP estimate of the latent variables

$$\hat{\mathbf{x}} = \arg\max_{\mathbf{x}} p(\mathbf{x}|\mathbf{y}) \tag{4}$$

and the MAP estimate of the hyperparameters,

$$\hat{\xi} = \arg\max_{\xi} p(\xi|\mathbf{y}) \,. \tag{5}$$

The following section discusses the criteria for and relationship between the two types of variational representation. In §4, we discuss algorithms for each problem based on the two types of variational representations, and determine when these are equivalent. We also discuss the approximation of the likelihood $p(\mathbf{y}; \mathbf{A})$ using the ensemble learning or VB method, which approximates the posterior $p(\mathbf{x}, \xi|\mathbf{y})$ by a factorial density $q(\mathbf{x}|\mathbf{y})q(\xi|\mathbf{y})$. We show that the ensemble method is equivalent to the hyperparameter MAP method.

## 3 Variational representations of super-Gaussian densities

In this section we discuss the criteria for the convex and integral type representations.

### 3.1 Convex variational bounds

We wish to determine when a symmetric, unimodal density $p(x)$ can be represented in the form (2) for some function $\varphi(\xi)$. Equivalently, when,

$$-\log p(x) = -\sup_{\xi > 0} \log \mathcal{N}(x; 0, \xi^{-1})\varphi(\xi) = \inf_{\xi > 0} \tfrac{1}{2} x^2 \xi - \log \xi^{\frac{1}{2}} \varphi(\xi)$$

for all $x > 0$. The last formula says that $-\log p(\sqrt{x})$ is the concave conjugate of (the closure of the convex hull of) the function, $\log \xi^{\frac{1}{2}} \varphi(\xi)$ [27, §12]. This is possible if and only if $-\log p(\sqrt{x})$ is closed, increasing and concave on $(0, \infty)$. Thus we have the following.

**Theorem 1.** *A symmetric probability density $p(x) \equiv \exp(-g(x^2))$ can be represented in the convex variational form,*

$$p(x) = \sup_{\xi > 0} \mathcal{N}(x; 0, \xi^{-1}) \, \varphi(\xi)$$

*if and only if $g(x) \equiv -\log p(\sqrt{x})$ is increasing and concave on $(0, \infty)$. In this case we can use the function,*

$$\varphi(\xi) = \sqrt{2\pi/\xi} \, \exp\big(g^*(\xi/2)\big),$$

*where $g^*$ is the concave conjugate of $g$.*

Examples of densities satisfying this criterion include: (i) Generalized Gaussian $\propto \exp(-|x|^\beta)$, $0 < \beta \leq 2$, (ii) Logistic $\propto 1/\cosh^2(x/2)$, (iii) Student's $t \propto (1 + x^2/\nu)^{-(\nu+1)/2}$, $\nu > 0$, and (iv) symmetric $\alpha$-stable densities (having characteristic function $\exp(-|\omega|^\alpha)$, $0 < \alpha \leq 2$).

The convex variational representation motivates the following definition.

**Definition 1.** *A symmetric probability density $p(x)$ is **strongly super-gaussian** if $p(\sqrt{x})$ is log-convex on $(0, \infty)$, and **strongly sub-gaussian** if $p(\sqrt{x})$ is log-concave on $(0, \infty)$.*

An equivalent definition is given in [5, pp. 60-61], which defines $p(x) = \exp(-f(x))$ to be sub-gaussian (super-gaussian) if $f'(x)/x$ is increasing (decreasing) on $(0, \infty)$. This condition is equivalent to $f(x) = g(x^2)$ with $g$ concave, i.e. $g'$ decreasing. The property of being strongly sub- or super-gaussian is independent of scale.

## 3.2 Scale mixtures

We now wish to determine when a probability density $p(x)$ can be represented in the form (3) for some $\mu(\xi)$ non-decreasing on $(0, \infty)$. A fundamental result dealing with integral representations was given by Bernstein and Widder (see [31]). It uses the following definition.

**Definition 1.** *A function $f(x)$ is **completely monotonic** on $(a, b)$ if,*

$$(-1)^n f^{(n)}(x) \geq 0, \quad n = 0, 1, \ldots$$

*for every $x \in (a, b)$.*

That is, $f(x)$ is completely monotonic if it is positive, decreasing, convex, and so on. Bernstein's theorem [31, Thm. 12b] states:

**Theorem 2.** *A necessary and sufficient condition that $p(x)$ should be completely monotonic on $(0, \infty)$ is that,*

$$p(x) = \int_0^\infty e^{-tx} d\alpha(t),$$

*where $\alpha(t)$ is non-decreasing on $(0, \infty)$.*

Thus for $p(x)$ to be a Gaussian scale mixture,

$$p(x) = e^{-f(x)} = e^{-g(x^2)} = \int_0^\infty e^{-\frac{1}{2}tx^2} d\alpha(t),$$

a necessary and sufficient condition is that $p(\sqrt{x}) = e^{-g(x)}$ be completely monotonic for $0 < x < \infty$, and we have the following (see also [19, 1]),

**Theorem 3.** *A function $p(x)$ can be represented as a Gaussian scale mixture if and only if $p(\sqrt{x})$ is completely monotonic on $(0, \infty)$.*

### 3.3 Relationship between convex and integral type representations

We now consider the relationship between the convex and integral types of variational representation. Let $p(x) = \exp(-g(x^2))$. We have seen that $p(x)$ can be represented in the form (2) if and only if $g(x)$ is symmetric and concave on $(0, \infty)$. And we have seen that $p(x)$ can be represented in the form (3) if and only if $p(\sqrt{x}) = \exp(-g(x))$ is completely monotonic. We shall consider now whether or not complete monotonicity of $p(\sqrt{x})$ implies the concavity of $g(x) = -\log p(\sqrt{x})$, that is whether representability in the integral form implies representability in the convex form.

Complete monotonicity of a function $q(x)$ implies that $q \geq 0$, $q' \leq 0$, $q'' \geq 0$, etc. For example, if $p(\sqrt{x})$ is completely monotonic, then,

$$\frac{d^2}{dx^2}\, p(\sqrt{x}) = \frac{d^2}{dx^2}\, e^{-g(x)} = e^{-g(x)}\big(g'(x)^2 - g''(x)\big) \geq 0\,.$$

Thus if $g'' \leq 0$, then $p(\sqrt{x})$ is convex, but the converse does not necessarily hold. That is, concavity of $g$ does not follow from convexity of $p(\sqrt{x})$, as the latter only requires that $g'' \leq g'^{\,2}$.

Concavity of $g$ does follow however from the complete monotonicity of $p(\sqrt{x})$. For example, we can use the following result [8, §3.5.2].

**Theorem 4.** *If the functions $f_t(x)$, $t \in \mathcal{D}$, are convex, then $\int_{\mathcal{D}} e^{f_t(x)} dt$ is convex.*

Thus completely monotonic functions, being scale mixtures of the log convex function $e^{-x}$ by Theorem 2, are also log convex. We thus see that *any function representable in the integral variational form* (3) *is also representable in the convex variational form* (2).

In fact, a stronger result holds. The following theorem [7, Thm. 4.1.5] establishes the equivalence between $q(x)$ and $g'(x) = d/dx - \log q(x)$ in terms of complete monotonicity.

**Theorem 5.** *If $g(x) > 0$, then $e^{-ug(x)}$ is completely monotonic for every $u > 0$, if and only if $g'(x)$ is completely monotonic.*

In particular, it holds that $q(x) \equiv p(\sqrt{x}) = \exp(-g(x))$ is convex only if $g''(x) \leq 0$.

To summarize, let $p(x) = e^{-g(x^2)}$. If $g$ is increasing and concave for $x > 0$, then $p(x)$ admits the convex type of variational representation (2). If, in addition, the higher derivatives satisfy $g^{(3)}(x) \geq 0$, $g^{(4)}(x) \leq 0$, $g^{(5)}(x) \geq 0$, etc., then $p(x)$ also admits the Gaussian scale mixture representation (3).

## 4 General equivalences among Variational methods

### 4.1 MAP estimation of components

Consider first the MAP estimate of the latent variables (4).

#### 4.1.1 Component MAP – Integral case

Following [10][1], consider an EM algorithm to estimate $\mathbf{x}$ when the $p(x_i)$ are independent Gaussian scale mixtures as in (3). Differentiating inside the integral gives,

$$\begin{aligned}
p'(x) &= \frac{d}{dx} \int_0^\infty p(x|\xi)p(\xi)d\xi = -\int_0^\infty \xi\, x\, p(x, \xi)\, d\xi \\
&= -x p(x) \int_0^\infty \xi p(\xi|x)\, d\xi\,.
\end{aligned}$$

Thus, with $p(x) \equiv \exp(-f(x))$, we see that,

$$E(\xi_i|x_i) = \int_0^\infty \xi_i p(\xi_i|x_i)\, d\xi_i = -\frac{p'(x_i)}{x_i p(x_i)} = \frac{f'(x_i)}{x_i}\,. \tag{6}$$

The EM algorithm alternates setting $\hat{\xi}_i$ to the posterior mean, $E(\xi_i|x_i) = f'(x_i)/x_i$, and setting $\mathbf{x}$ to minimize,

$$-\log p(\mathbf{y}|\mathbf{x})p(\mathbf{x}|\hat{\xi}) = \tfrac{1}{2}\mathbf{x}^T\mathbf{A}^T\boldsymbol{\Sigma}_\nu^{-1}\mathbf{A}\mathbf{x} - \mathbf{y}^T\boldsymbol{\Sigma}_\nu^{-1}\mathbf{A}\mathbf{x} + \tfrac{1}{2}\mathbf{x}^T\boldsymbol{\Lambda}\mathbf{x} + \text{const.}, \tag{7}$$

where $\boldsymbol{\Lambda} = \mathrm{diag}(\hat{\xi})^{-1}$. At iteration $k$, we put $\xi_i^k = f'(x_i^k)/x_i^k$, and $\boldsymbol{\Lambda}^k = \mathrm{diag}(\xi^k)^{-1}$, and

$$\mathbf{x}^{k+1} = \boldsymbol{\Lambda}^k\mathbf{A}^T(\mathbf{A}\boldsymbol{\Lambda}^k\mathbf{A}^T + \boldsymbol{\Sigma}_\nu)^{-1}\mathbf{y}\,.$$

### 4.1.2 Component MAP – Convex case

Again consider the MAP estimate of $\mathbf{x}$. For strongly super-gaussian priors, $p(x_i)$, we have,

$$\arg\max_{\mathbf{x}} p(\mathbf{x}|\mathbf{y}) = \arg\max_{\mathbf{x}} p(\mathbf{y}|\mathbf{x})p(\mathbf{x}) = \arg\max_{\mathbf{x}} \max_{\xi} p(\mathbf{y}|\mathbf{x})p(\mathbf{x};\xi)\varphi(\xi)$$

Now since,

$$-\log p(\mathbf{y}|\mathbf{x})p(\mathbf{x};\xi)\varphi(\xi) = \tfrac{1}{2}\mathbf{x}^T\mathbf{A}^T\boldsymbol{\Sigma}_\nu^{-1}\mathbf{A}\mathbf{x} - \mathbf{y}^T\boldsymbol{\Sigma}_\nu^{-1}\mathbf{A}\mathbf{x} + \sum_{i=1}^n \tfrac{1}{2}x_i^2\xi_i - g^*(\xi_i/2)\,,$$

the MAP estimate can be improved iteratively by alternately maximizing $\mathbf{x}$ and $\xi$,

$$\xi_i^k = 2\,g^{*\prime-1}(x_i^{k\,2}) = 2\,g'(x_i^{k\,2}) = \frac{f'(x_i^k)}{x_i^k}\,, \tag{8}$$

with $\mathbf{x}$ updated as in §4.1.1. We thus see that this algorithm is equivalent to the MAP algorithm derived in §4.1.1 for Gaussian scale mixtures. That is, for direct MAP estimation of latent variable $\mathbf{x}$, the EM Gaussian scale mixture method and the variational bounding method yield the same algorithm.

This algorithm has also been derived in the image restoration literature [12] as the "half-quadratic" algorithm, and it is the basis for the FOCUSS algorithms derived in [26, 25]. The regression algorithm given in [11] for the particular cases of Laplacian and Jeffrey's priors is based on the theory in §4.1.1, and is in fact equivalent to the FOCUSS algorithm derived in [26].

## 4.2 MAP estimate of variational parameters

Now consider MAP estimation of the (random) variational hyperparameters $\xi$.

### 4.2.1 Hyperparameter MAP – Integral case

Consider an EM algorithm to find the MAP estimate of the hyperparameters $\xi$ in the integral representation (Gaussian scale mixture) case, where the latent variables $\mathbf{x}$ are hidden. For the complete likelihood, we have,

$$p(\xi,\mathbf{x}|\mathbf{y}) \propto p(\mathbf{y}|\mathbf{x},\xi)p(\mathbf{x}|\xi)p(\xi) = p(\mathbf{y}|\mathbf{x})p(\mathbf{x}|\xi)p(\xi)\,.$$

The function to be minimized over $\xi$ is then,

$$\left\langle -\log p(\mathbf{x}|\xi)p(\xi)\right\rangle_{\mathbf{x}} = \sum_i \tfrac{1}{2}\left\langle x_i^2\right\rangle\xi_i - \log\sqrt{\xi_i}\,p(\xi_i) + \text{const.} \tag{9}$$

If we define $h(\xi) \equiv \log\sqrt{\xi_i}\,p(\xi_i)$, and assume that this function is concave, then the optimal value of $\xi$ is given by,

$$\xi_i = h^{*\prime}\left(\tfrac{1}{2}\left\langle x_i^2\right\rangle\right)\,.$$

This algorithm converges to a local maximum of $p(\xi|\mathbf{y})$, $\hat{\xi}$, which then yields an estimate of $\mathbf{x}$ by taking $\hat{\mathbf{x}} = E(\mathbf{x}|\mathbf{y},\hat{\xi})$. Alternative algorithms result from using this method to find the MAP estimate of different functions of the scale random variable $\xi$.

### 4.2.2 Hyperparameter MAP – Convex case

In the convex representation, the $\xi$ parameters do not actually represent a probabilistic quantity, but rather arise as parameters in a variational inequality. Specifically, we write,

$$
\begin{aligned}
p(\mathbf{y}) \;&=\; \int p(\mathbf{y}, \mathbf{x})\, d\mathbf{x} \;=\; \int \max_{\xi} p(\mathbf{y}|\mathbf{x})\, p(\mathbf{x}|\xi)\, \varphi(\xi)\, d\mathbf{x} \\
&\geq\; \max_{\xi} \int p(\mathbf{y}|\mathbf{x})\, p(\mathbf{x}|\xi)\, \varphi(\xi)\, d\mathbf{x} \\
&=\; \max_{\xi} \mathcal{N}\!\left(\mathbf{y}; \mathbf{0}, \mathbf{A\Lambda A}^{T} + \mathbf{\Sigma}_{\nu}\right) \varphi(\xi)\,.
\end{aligned}
$$

Now we define the function,

$$
\tilde{p}(\mathbf{y}; \xi) \;\equiv\; \mathcal{N}\!\left(\mathbf{y}; \mathbf{0}, \mathbf{A\Lambda A}^{T} + \mathbf{\Sigma}_{\nu}\right) \varphi(\xi)
$$

and try to find $\hat{\xi} = \arg\max \tilde{p}(\mathbf{y}; \xi)$. We maximize $\tilde{p}$ by EM, marginalizing over $\mathbf{x}$,

$$
\tilde{p}(\mathbf{y}; \xi) = \int p(\mathbf{y}|\mathbf{x})\, p(\mathbf{x}|\xi)\, \varphi(\xi)\, d\mathbf{x}\,.
$$

The algorithm is then equivalent to that in §4.1.2 except that the expectation is taken of $x^2$ as the E step, and the diagonal weighting matrix becomes,

$$
\xi_i \;=\; \frac{f'(\sigma_i)}{\sigma_i}\,,
$$

where $\sigma_i = \sqrt{E\left(x_i^2 | \mathbf{y}; \xi_i\right)}$. Although $\tilde{p}$ is not a true probability density function, the proof of convergence for EM does not assume unit normalization. This theory is the basis for the algorithm presented in [14] for the particular case of a Laplacian prior (where in addition $\mathbf{A}$ in the model (1) is updated according to the standard EM update.)

### 4.3 Ensemble learning

In the ensemble learning approach (also Variational Bayes [4, 3, 6]) the idea is to find the approximate separable posterior that minimizes the KL divergence from the true posterior, using the following decomposition of the log likelihood,

$$
\begin{aligned}
\log p(\mathbf{y}) \;&=\; \int q(\mathbf{z}|\mathbf{y}) \log \frac{p(\mathbf{z}, \mathbf{y})}{q(\mathbf{z}|\mathbf{y})}\, d\mathbf{z} + D\big(q(\mathbf{z}|\mathbf{y}) \,\|\, p(\mathbf{z}|\mathbf{y})\big) \\
&\equiv\; -F(q)\,+\,D(q\|p)\,.
\end{aligned}
$$

The term $F(q)$ is commonly called the *variational free energy* [29, 24]. Minimizing the $F$ over $q$ is equivalent to minimizing $D$ over $q$. The posterior approximating distribution is taken to be factorial,

$$
q(\mathbf{z}|\mathbf{y}) = q(\mathbf{x}, \xi|\mathbf{y}) = q(\mathbf{x}|\mathbf{y})q(\xi|\mathbf{y})\,.
$$

For fixed $q(\xi|\mathbf{y})$, the free energy $F$ is given by,

$$
-\iint q(\mathbf{x}|\mathbf{y})q(\xi|\mathbf{y}) \log \frac{p(\mathbf{x}, \xi|\mathbf{y})}{q(\mathbf{x}|\mathbf{y})q(\xi|\mathbf{y})}\, d\xi\, d\mathbf{x} \;=\; D\Big(q(\mathbf{x}|\mathbf{y}) \,\big\|\, e^{\langle \log p(\mathbf{x},\xi|\mathbf{y})\rangle_\xi}\Big) + \text{const.,}
\tag{10}
$$

where $\langle \cdot \rangle_\xi$ denotes expectation with respect to $q(\xi|\mathbf{y})$, and the constant is the entropy, $H\big(q(\xi|\mathbf{y})\big)$. The minimum of the KL divergence in (10) is attained if and only if

$$
q(\mathbf{x}|\mathbf{y}) \propto \exp\big\langle \log p(\mathbf{x}, \xi|\mathbf{y})\big\rangle_\xi \propto p(\mathbf{y}|\mathbf{x}) \exp\big\langle \log p(\mathbf{x}|\xi)\big\rangle_\xi
$$

almost surely. An identical derivation yields the optimal

$$
q(\xi|\mathbf{y}) \propto \exp\big\langle \log p(\mathbf{x}, \xi|\mathbf{y})\big\rangle_\mathbf{x} \propto p(\xi) \exp\big\langle \log p(\mathbf{x}|\xi)\big\rangle_\mathbf{x}
$$

when $q(\mathbf{x}|\mathbf{y})$ is fixed. The ensemble (or VB) algorithm consists of alternately updating the parameters of these approximating marginal distributions.

In the linear model with Gaussian scale mixture latent variables, the complete likelihood is again,

$$p(\mathbf{y}, \mathbf{x}, \xi) \;=\; p(\mathbf{y}|\mathbf{x})p(\mathbf{x}|\xi)p(\xi) \,.$$

The optimal approximate posteriors are given by,

$$q(\mathbf{x}|\mathbf{y}) \;=\; \mathcal{N}(\mathbf{x}; \mu_{\mathbf{x}|\mathbf{y}}, \Sigma_{\mathbf{x}|\mathbf{y}})\,, \qquad q(\xi_i|\mathbf{y}) \;=\; p\left(\xi_i \,\big|\, x_i = \langle x_i^2\rangle^{1/2}\right),$$

where, letting $\boldsymbol{\Lambda} = \mathrm{diag}(\langle\xi\rangle)^{-1}$, the posterior moments are given by,

$$
\begin{aligned}
\mu_{\mathbf{x}|\mathbf{y}} &\equiv \boldsymbol{\Lambda}\mathbf{A}^T(\mathbf{A}\boldsymbol{\Lambda}\mathbf{A}^T + \boldsymbol{\Sigma}_\nu)^{-1}\mathbf{y} \\
\Sigma_{\mathbf{x}|\mathbf{y}} &\equiv (\mathbf{A}^T\boldsymbol{\Sigma}_\nu^{-1}\mathbf{A} + \boldsymbol{\Lambda}^{-1})^{-1} \;=\; \boldsymbol{\Lambda} - \boldsymbol{\Lambda}\mathbf{A}^T(\mathbf{A}\boldsymbol{\Lambda}\mathbf{A}^T + \boldsymbol{\Sigma}_\nu)^{-1}\mathbf{A}\boldsymbol{\Lambda} \,.
\end{aligned}
$$

The only relevant fact about $q(\xi|\mathbf{y})$ that we need is $\langle\xi\rangle$, for which we have, using (6),

$$\langle\xi_i\rangle \;=\; \int \xi_i q(\xi_i|\mathbf{y})\, d\xi_i \;=\; \int \xi_i p\left(\xi_i \,\big|\, x_i = \langle x_i^2\rangle^{1/2}\right) d\xi_i \;=\; \frac{f'(\sigma_i)}{\sigma_i}\,,$$

where $\sigma_i = \sqrt{E\left(x_i^2|\mathbf{y}; \xi_i\right)}$. We thus see that the ensemble learning algorithm is equivalent to the approximate hyperparameter MAP algorithm of §4.2.2. Note also that this shows that the VB methods can be applied to any Gaussian scale mixture density, using only the form of the latent variable prior $p(x)$, without needing the marginal hyperprior $p(\xi)$ in closed form. This is particularly important in the case of the Generalized Gaussian and Logistic densities, whose scale parameter densities are $\alpha$-Stable and Kolmogorov [1] respectively.

## 5  Conclusion

In this paper, we have discussed criteria for variational representations of non-Gaussian latent variables, and derived general variational EM algorithms based on these representations. We have shown a general equivalence between the two representations in MAP estimation taking hyperparameters as hidden, and we have shown the general equivalence between the variational convex approximate MAP estimate of hyperparameters and the ensemble learning or VB method.

## Footnotes

[1]In [10], the $x_i$ in (1) are actually estimated as non-random parameters, with the noise $\nu$ being non-gaussian, but the underlying theory is essentially the same.

## References

[1] D. F. Andrews and C. L. Mallows. Scale mixtures of normal distributions. *J. Roy. Statist. Soc. Ser. B*, 36:99–102, 1974.

[2] H. Attias. Independent factor analysis. *Neural Computation*, 11:803–851, 1999.

[3] H. Attias. A variational Bayesian framework for graphical models. In *Advances in Neural Information Processing Systems 12*. MIT Press, 2000.

[4] M. J. Beal and Z. Ghahrarmani. The variational Bayesian EM algorithm for incomplete data: with application to scoring graphical model structures. In *Bayesian Statistics 7*, pages 453–464. University of Oxford Press, 2002.

[5] A. Benveniste, M. Métivier, and P. Priouret. *Adaptive algorithms and stochastic approximations*. Springer-Verlag, 1990.

[6] C. M. Bishop and M. E. Tipping. Variational relevance vector machines. In C. Boutilier and M. Goldszmidt, editors, *Proceedings of the 16th Conference on Uncertainty in Artificial Intelligence*, pages 46–53. Morgan Kaufmann, 2000.

[7] S. Bochner. *Harmonic analysis and the theory of probability*. University of California Press, Berkeley and Los Angeles, 1960.

[8] S. Boyd and L. Vandenberghe. *Convex Optimization*. Cambridge University Press, 2004.

[9] A. P. Dempster, N. M. Laird, and D. B. Rubin. Maximum likelihood from incomplete data via the EM algorithm. *Journal of the Royal Statistical Society, Series B*, 39:1–38, 1977.

[10] A. P. Dempster, N. M. Laird, and D. B. Rubin. Iteratively reweighted least squares for linear regression when errors are Normal/Independent distributed. In P. R. Krishnaiah, editor, *Multivariate Analysis V*, pages 35–57. North Holland Publishing Company, 1980.

[11] M. Figueiredo. Adaptive sparseness using Jeffreys prior. In T. G. Dietterich, S. Becker, and Z. Ghahramani, editors, *Advances in Neural Information Processing Systems 14*, Cambridge, MA, 2002. MIT Press.

[12] D. Geman and G. Reynolds. Constrained restoration and the recovery of discontinuities. *IEEE Trans. Pattern Analysis and Machine Intelligence*, 14(3):367–383, 1992.

[13] Z. Ghahramani and M. J. Beal. Variational inference for Bayesian mixtures of factor analysers. In *Advances in Neural Information Processing Systems 12*. MIT Press, 2000.

[14] M. Girolami. A variational method for learning sparse and overcomplete representations. *Neural Computation*, 13:2517–2532, 2001.

[15] A. Honkela and H. Valpola. Unsupervised variational Bayesian learning of nonlinear models. In *Advances in Neural Information Processing Systems 17*. MIT Press, 2005.

[16] T. S. Jaakkola. *Variational Methods for Inference and Estimation in Graphical Models*. PhD thesis, Massachusetts Institute of Technology, 1997.

[17] T. S. Jaakkola and M. I. Jordan. A variational approach to Bayesian logistic regression models and their extensions. In *Proceedings of the 1997 Conference on Artificial Intelligence and Statistics*, 1997.

[18] M. I. Jordan, Z. Ghahramani, T. S. Jaakkola, and L. K. Saul. An introduction to variational methods for graphical models. In M. I. Jordan, editor, *Learning in Graphical Models*. Kluwer Academic Publishers, 1998.

[19] J. Keilson and F. W. Steutel. Mixtures of distributions, moment inequalities, and measures of exponentiality and Normality. *The Annals of Probability*, 2:112–130, 1974.

[20] H. Lappalainen. Ensemble learning for independent component analysis. In *Proceedings of the First International Workshop on Independent Component Analysis*, 1999.

[21] D. J. C. MacKay. Bayesian interpolation. *Neural Computation*, 4(3):415–447, 1992.

[22] D. J. C. MacKay. Ensemble learning and evidence maximization. Unpublished manuscript, 1995.

[23] D. J. C. Mackay. Comparison of approximate methods for handling hyperparameters. *Neural Computation*, 11(5):1035–1068, 1999.

[24] R. M. Neal and G. E. Hinton. A view of the EM algorithm that justifies incremental, sparse, and other variants. In M. I. Jordan, editor, *Learning in Graphical Models*, pages 355–368. Kluwer, 1998.

[25] B. D. Rao, K. Engan, S. F. Cotter, J. Palmer, and K. Kreutz-Delgado. Subset selection in noise based on diversity measure minimization. *IEEE Trans. Signal Processing*, 51(3), 2003.

[26] B. D. Rao and I. F. Gorodnitsky. Sparse signal reconstruction from limited data using FOCUSS: a re-weighted minimum norm algorithm. *IEEE Trans. Signal Processing*, 45:600–616, 1997.

[27] R. T. Rockafellar. *Convex Analysis*. Princeton, 1970.

[28] Sam Roweis and Zoubin Ghahramani. A unifying review of linear gaussian models. *Neural Computation*, 11(5):305–345, 1999.

[29] L. K. Saul, T. S. Jaakkola, and M. I. Jordan. Mean field theory for sigmoid belief networks. *Journal of Artificial Intelligence Research*, 4:61–76, 1996.

[30] M. E. Tipping. Sparse Bayesian learning and the Relevance Vector Machine. *Journal of Machine Learning Research*, 1:211–244, 2001.

[31] D. V. Widder. *The Laplace Transform*. Princeton University Press, 1946.

[32] D. Wipf, J. Palmer, and B. Rao. Perspectives on sparse bayesian learning. In S. Thrun, L. Saul, and B. Schölkopf, editors, *Advances in Neural Information Processing Systems 16*, Cambridge, MA, 2003. MIT Press.
